# Decoding V1 Neuronal Activity using Particle Filtering with Volterra Kernels

**Ryan Kelly**
Center for the Neural Basis of Cognition
Carnegie-Mellon University
Pittsburgh, PA 15213
rkelly@cs.cmu.edu

**Tai Sing Lee**
Center for the Neural Basis of Cognition
Carnegie-Mellon University
Pittsburgh, PA 15213
tai@cnbc.cmu.edu

## Abstract

Decoding is a strategy that allows us to assess the amount of information neurons can provide about certain aspects of the visual scene. In this study, we develop a method based on Bayesian sequential updating and the particle filtering algorithm to decode the activity of V1 neurons in awake monkeys. A distinction in our method is the use of Volterra kernels to filter the particles, which live in a high dimensional space. This parametric Bayesian decoding scheme is compared to the optimal linear decoder and is shown to work consistently better than the linear optimal decoder. Interestingly, our results suggest that for decoding in real time, spike trains of as few as 10 independent but similar neurons would be sufficient for decoding a critical scene variable in a particular class of visual stimuli. The reconstructed variable can predict the neural activity about as well as the actual signal with respect to the Volterra kernels.

## 1 Introduction

Cells in the primary visual cortex perform nonlinear operations on visual stimuli. This nonlinearity introduces ambiguity in the response of the neurons. Given a neuronal response, an optimal linear decoder cannot accurately reconstruct the visual stimulus due to nonlinearities. Is there a strategy to resolve this ambiguity and recover the information that is encoded in the response of these neurons?

Bayesian decoding schemes, which are nonlinear, might be useful in this context . Bayesian sequential updating or belief propagation, implemented in the form of particle filtering, has recently been used in estimating the hand trajectories of monkeys based on M1 neuron's responses [4] and the location of a rat based on the responses of the place cells in the hippocampus[3]. However, linear methods have been shown to be quite adequate for decoding LGN, motor cortical, or hippocampal place cells' signals using population vectors or the optimal linear decoder [10, 5, 8]. Bayesian methods, with proper probability model assumptions, could work better than the linear methods, but they apparently are not critical to solving those problems. These methods may be more useful or important in the decoding of nonlinear visual neuronal responses. Here, we implement an algorithm based on Bayesian sequential updating in the form particle filtering to decode nonlinear visual neurons in awake behaving monkeys. The strategy is similar to the one used by Brown et

al. [2] and Brockwell et al. [1] in their decoding of hippocampus place neurons or M1 neurons, except that we introduced the use of Volterra kernels [6, 7, 9] to filter the hypothesis particle to generate feedback messages. The Volterra kernels integrate information from the previous 200 ms. This window allows us to backtrack and update the hypotheses within a 200 ms window, so the hypothesis space does not grow beyond 200ms for lengthy signals. We demonstrated that this method is feasible practically and indeed useful for decoding a temporal variable in the stimulus input based on cells' responses and that it succeeds even when the optimal linear decoder fails.

## 2  The Approach

Our objective is to infer the time-series of a scene variable based on the ongoing response of one or a set of visual neurons. A hypothesis particle is then the entire history of the scene variable of interest up to the present time $t$, i.e. $(x_1, x_2, \ldots, x_t)$ given the observed neuronal activity $(y_1, y_2, \ldots, y_t)$. A key feature of our algorithm is the use of a decoded or estimated hypothesis to predict the response of the neurons at the next time step. The premise is that the scene variable we are inferring is sufficient to predict the activity of the neuron. Since visual neurons have a temporal receptive field and typically integrate information from the past 100-200 ms to produce a response, we cannot make the Markovian assumption made in other Bayesian decoding studies [1, 2, 3, 4]. Instead, we will use the receptive field (kernel) to filter each hypothesis particle to generate a prediction of the neural response. We propose to use the Volterra kernels, which have been used in previous studies [6, 7, 9] to characterize the transfer function or receptive field of a neuron, to filter the hypothesis $(\hat{x}_t, \ldots \hat{x}_1)$. The predicted response of the neuron according to the kernels is based on the stimulus in the last 200 ms, optionally incorporating some lag which we eliminated by shifting the response forward 40 ms in time to compensate for the 40 ms the visual signal required to travel from the retina to V1.

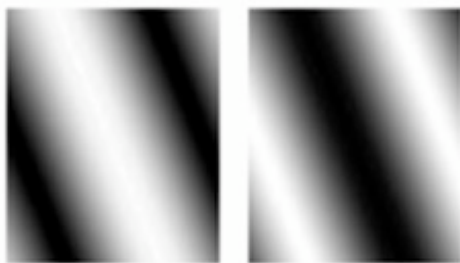

Figure 1: Two sample sinewave gratings.

Ongoing observation of the activity of neurons is compared to the predicted response or proposal to yield a likelihood measure. The likelihood measure of each hypothesis particle is proportional to how close the hypothesis's predicted response is to the actual observed neural response. As all the existing hypotheses are weighted by their likelihood measures, the posterior distribution of the hypothesis is effectively resampled. The hypotheses that tend to generate incorrect proposals will die off over time. Conversely, the hypotheses that give predicted responses close to the actual response values will not only be kept alive, but also be allowed to give birth to offspring particles in its vicinity in the hypothesis space, allowing the algorithm to zoom in to the correct hypothesis more precisely.

After weighting, resampling and reproducing, the hypothesis particles are propagated forward according to the prior statistical distribution on how the scene variable tends to progress. That is, $p(x_t|x_{t-1})$ yields a proposed hypothesis about the stimulus at time $t + 1$ based on the existing hypothesis which is defined at $t$ and earlier times. These hypotheses are then filtered though the Volterra kernels to predict the distribution $p(y_t|x_{t-200,\ldots,t-1})$, thus completing the loop. The entire flow-chart of our inference system is shown in Figure 4. Each step is described in detail below.

# 3 Neurophysiological Experiment

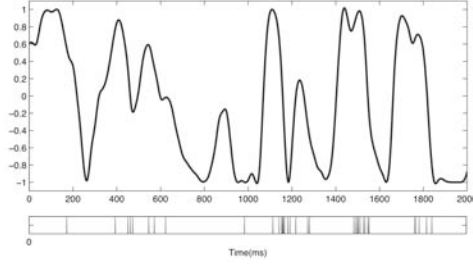

Figure 2: A sample time series of the scene variable, with a sample spike train below.

We applied the ideas above to the data obtained by the following experiment. This experiment sought to understand the encoding and decoding of temporal visual information by V1 neurons. In each experimental session, a movie (2.2 seconds per trial) of a sinewave grating stimulus was presented while the monkey had to maintain fixation on a spot within a $0.8^o \times 0.8^o$ window. The sinewave grating was constrained to move along one dimension in a direction perpendicular to the grating with a step size in phase drawn from a random pink noise distribution which follow a 1/f power spectrum in the Fourier domain, approximating the statistical correlational structures in natural temporal stimuli. To ensure continuity of the input signals we took the cosine of the phase, which is related to the image intensity value at a local area within the receptive field. In decoding the cos(phase), a hidden variable, was the scene variable inferred. A sample stimulus is given in Figure 2. This scene variable, through the Volterra kernel procedure, can predict the neural responses of this class of stimulus reasonably well.

400 trials of different sequences were presented. The known pair sequences of stimulus and response in these trials were used to estimate the Volterra kernels by correlating the input $x$ with the neural response $y$. In addition, one particular stimulus sequence is repeated 60-80 trials to obtain a PSTH, which is smoothed with a 10 ms window to give an estimate of the instantaneous firing rate. In our decoding work, we take the PSTH as input to our algorithm; this is considered equivalent to assuming simultaneous access to a number of identical, independent neurons. When the neurons are different, a kernel derivation for each neuron is necessary.

# 4 Volterra Kernels

Volterra kernels have been used to characterize a cell's transfer function. With Volterra kernels with memory length $L$, the response $y_t$ can be predicted by convolution of the kernels with the input $x_t$,

$$y(t) = y_t = h_o + \sum_{\tau=1}^{L} h_\tau x_{t-\tau} + \sum_{\tau_2}^{L} \sum_{\tau_1}^{L} h_{\tau_1,\tau_2} x_{t-\tau_1} x_{t-\tau_2},$$

where $h_0$ corresponds to the mean firing rate, $h_\tau$ is the first order kernel and $h_{\tau_1,\tau_2}$ the second order kernel. We restrict all $\tau$'s to be positive, so we only consider causal filters. This equation is easily expressed in matrix form as $Y = XH$, where time is now indexed by matrix row in $Y$ and $X$. $H$ contains the concatenation of the terms

$$[h_0 \ h_1 \ \cdots \ h_L \ h_{1,1} \ h_{1,2} \ \cdots \ h_{L,L}]',$$

and row $t$ of $X$ is similarly

$$[1 \ x_{t-1} \ \cdots \ x_{t-L} \ (x_{t-1} \ x_{t-1}) \ \cdots \ (x_{t-L} \ x_{t-L})]$$

The standard solution for this regression problem is $H = (X'X)^{-1} X'Y$. That is, the parameters of the kernels are derived using the regression technique by correlating the input and the output, and are compensated by the covariance in the input, i.e.

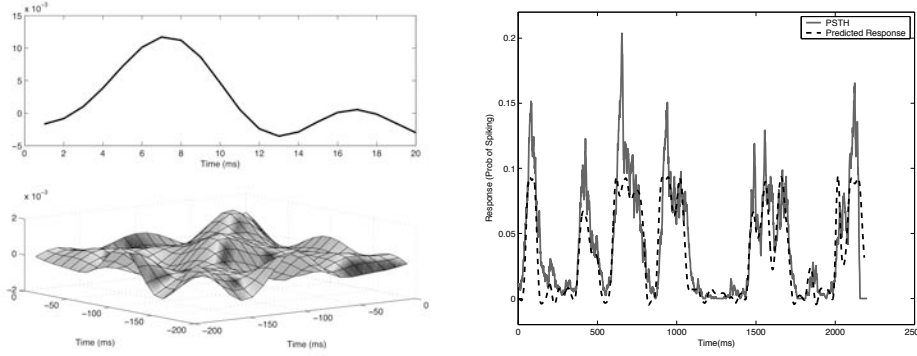

Figure 3: The first and second order Volterra kernels of a V1 cell (left) and a typical prediction of the neuronal response compared to the actual response (right).

$H = (X'X)^{-1}X'Y$. Because of the correlations in the input signal $x_t$, the matrix $(X'X)$ is ill conditioned. Instead of directly inverting this matrix, singular value decomposition can be used, as $USU' = X'X$ where $US^{-1}U' = (X'X)^{-1}$ and $S$ is a diagonal matrix. Only the first $n$ largest dimensions as ranked by their eigenvalue are included, where $n$ is chosen to account for $99\%$ of the variance in X [7].

Figure 3 depicts an example of the first and second order Volterra kernels and also shows a typical example of their accuracy in predicting the response PSTH $y_t$. For a majority of these neurons, the Volterra kernels recovered are capable of predicting the neural response to the input stimulus with a high level of accuracy. This observation forms the basis of success for our scheme of particle filtering.

## 5 Decoding Scheme

We apply Bayesian decoding to the problem of determining the visual stimulus variable $x_t$ for some time step $t$, given observations of the neural responses $(y_1, y_2, \ldots, y_t)$. The global flow of the algorithm is shown in Figure 4.

### 5.1 Particle Prediction

At each time step of of decoding scheme, we can now filter a hypothesis particle $(\hat{x}_1, \hat{x}_2, \ldots, \hat{x}_t)$ by the Volterra kernels to generate a prediction of the response of the neuron to test the validity of the hypothesis. $(y_1, y_2, \ldots, y_t)$ remains the observed neural activity of a V1 neuron up to time $t$, and $\hat{y}_t^i$ is the predicted neural activity at time $t$ based on hypothesis particle $i$. This gives us a set of predicted responses at time $t$, $\{\hat{y}_t^1, \hat{y}_t^2, \ldots, \hat{y}_t^N\}$, where the subscript is the particle index, and N is the number of particles.

### 5.2 Particle Resampling

The actual observed response of the neuron at time $t$ is compared to each particle's prediction as a means to evaluate the likelihood or fitness of the particle. If we assume $y_t$ is the average of spike trains from a single neuron in independent trials or the average firing rate of a population of independent neurons with identical tuning properties, then the resulting error distribution can be assumed to be a Gaussian distribution, with $\sigma$ representing the uncertainty of the predicted response given the correct values of the stimulus variable. The

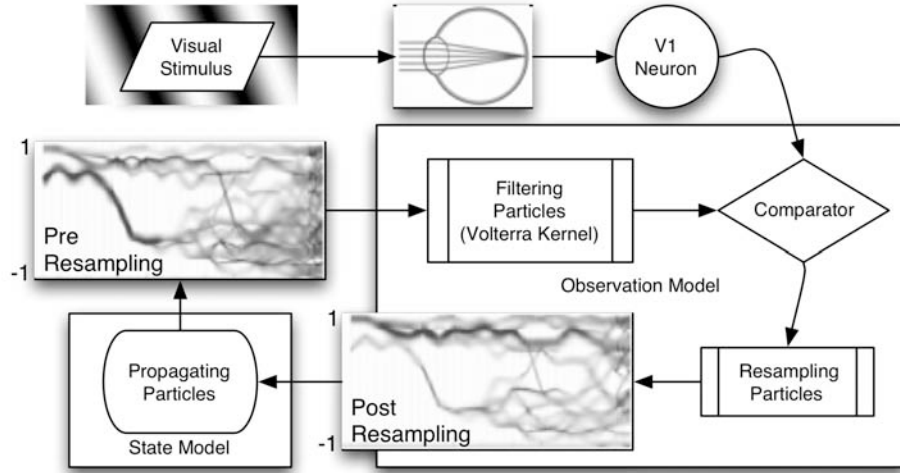

Figure 4: Flow chart of the PF decoding scheme. The effect of one resampling step is shown in the two graphs. Each graph shows the particles' (n=100) values during a trial over 200 ms. The thicknesses of the lines are proportional to the number of particles with the corresponding values. Notice the change in the distribution of particles after sampling. After the resampling there are many more particles concentrated around 1 instead of -1.

relative likelihood of an observation given each particle is then given by

$$p(y_t|\hat{x}_1^i, \ldots, \hat{x}_t^i) = \frac{e^{-(\hat{y}_t^i - y_t)^2/2\sigma^2}}{\sum_j e^{-(\hat{y}_t^j - y_t)^2/2\sigma^2}}.$$

All the particles together provide a representation of the particle-conditional distribution,

$$p(y_t|\hat{x}_t, \hat{x}_{t-1}, \ldots, \hat{x}_1).$$

This is used to resample the posterior distribution of the hypotheses based on all the observations up to time $t - 1$,

$$p(\hat{x}_t|y_1, y_2, \ldots y_t) \propto p(y_t|\hat{x}_t)p(\hat{x}_t|y_1, y_2, \ldots y_{t-1}),$$

to produce a current posterior distribution of the hypotheses.

### 5.3 Particle Propagation

The next step in the decoding scheme is to generate new value $\hat{x}_{t+1}$ and append it to the hypothesis particle

$$p(\hat{x}_{t+1}|y_1, y_2, \ldots y_t) = \int p(\hat{x}_{t+1}|\hat{x}_t)p(\hat{x}_t|y_1, y_2, \ldots y_t)dx_t,$$

where $p(\hat{x}_{t+1}|\hat{x}_t)$ is the state propagation model that provides the prior on how the stimulus changes over time. For the state propagation model used in this study, all initial positions for the stimulus are equally likely. The range of the stimulus (-1 to 1) is divided into 60 equally spaced intervals. A 60x60 probability table is constructed empirically from the training data stimuli, corresponding to a discrete version approximating the conditional prior above. Solving these priors analytically is difficult or even impossible. Besides, the

hypothesis space is enormous as there are 60 possible values at each time point, and information from a 200 ms window (20 time points at 10 ms intervals) is being integrated to predict $y_t$. The particle filtering algorithm is basically a way to approximate the distributions efficiently.

The algorithm consists of cycling through the above steps, i.e. particle prediction, particle resampling, and particle propagation. In summary,

1. Prediction step: Filter all particles by the Volterra kernels to generate the prediction of neural responses.

2. Resampling step: Compare actual neural response with the predicted response of each particle to assign a likelihood value to each particle. Resample (with replacement) the posterior distribution of the particles based on their likelihood.

3. Propagation step: Sample from the state model to randomly postulate a new stimulus value $\hat{x}_t$ for each particle and add this value to the end of the particle's sequence to obtain $(\hat{x}_1, \hat{x}_2, \ldots, \hat{x}_t)$.

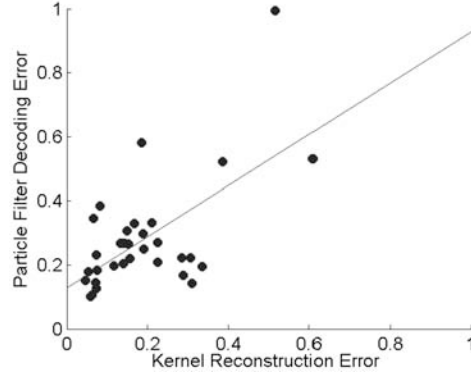

Figure 5: A scatter plot showing the least squares regression line for the data.

In the propagation step, the state model will move the stimulus in ways that it has typically been seen to move. In the prediction step, particles that predict a neural response close to the actual observed response will be highly valued and will likely be duplicated in the resampled set. Conversely, particles that predict a response which is not close to the actual response will not be highly valued and thus will likely be removed from the resultant set.

## 6 Results and Discussion

Let $x_t, x_{t-1}, \ldots, x_1$ be the inferred scene variable (cos(phase)). $s_k(t)$ is the binary spike response of a neuron during trial $k$. The instantaneous firing rate of the neuron is given by

$$y(t) = \frac{1}{m} \sum_k^m = 1 s_k(t)$$

where $m$ is the number of trials. In general, for cells that respond well to a stimulus, this first order and the second order kernel can predict the response well. Over all cells tested (n=33), the average error ratio $e_y$ in the energy of the actual response is 18.4%. Each of the cells was decoded using the particle filtering algorithm with 1000 particles. The average reconstruction error $e_x$ is 27.14%, and the best cell has 10% error. A correlation exists between the encoding and decoding errors across trials as shown in Figure 5.

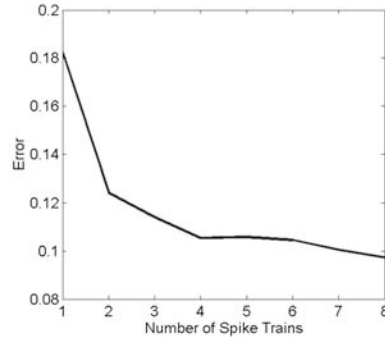

Figure 6: Reconstruction error when input PSTH is constructed from fewer trials. With 10 spike trains, the PF has almost achieved the minimum error possible for this cell.

$$e_y = \frac{\sum_t (\hat{y}_t - y_t)^2}{\sum_t y_t^2}, e_x = \frac{\sum_t (\hat{x}_t - x_t)^2}{\sum_t (x_t + 1)^2},$$

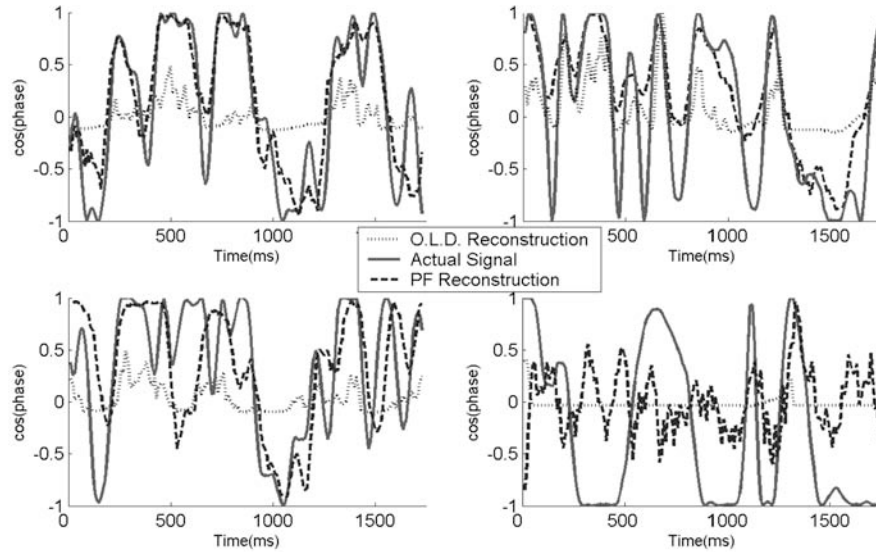

Figure 7: Particle filtering (PF) and optimal linear decoder (O.L.D.) reconstructions. The top left is the best PF reconstruction, and the bottom right is the worst out of all the cells tested.

$\sigma$ affects the rate at which the particle hypothesis space collapses around the correct solution. If $\sigma$ is too large, all particles will become equally likely, while if $\sigma$ is too small, only a few particles will survive each time step. Ideally, the particles will converge on a value for a number of time steps equal to the kernel's length. The optimal value for $\sigma$ was found empirically and was used in all reconstructions.

Figure 7 shows sample reconstructions of some good and bad cells. Decoding accuracy is limited by the performance of the Volterra kernel. When the kernel is unable to predict the neuronal response, particularly for cells that have low firing rates, any decoding scheme will suffer because of insufficient information. Thus the amount of error is correlated to the inability of the kernel in predicting neuronal responses. This idea is consistent with the error correlation between the particle filter and kernel in Figure 5. These cells do not provide enough relevant information about the visual stimulus in their spiking activities.

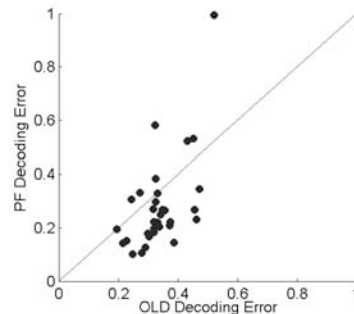

Figure 8: A scatter plot comparing the two decoding methods.

Figure 6 shows that reconstruction based on the PSTH constructed from as few as 5-10 spike trains can reach an accuracy not far from reconstruction based on the PSTH of 80 trials. This suggests that as few as 10 independent but similar cells recorded simultaneously might be sufficient for decoding this scene variable.

We find that the optimal linear decoder does not decode these cells well. The decoded output tends to follow the signal somewhat, but at a low amplitude as shown in Figure 7. The problem for the optimal linear decoder is that at any single moment in time it can only propose a single hypothesis, but there exist multiple signals that can produce the response.

The optimal linear decoder tends to average in these cases. The particle filter keeps alive many independent hypotheses and can thus choose the most likely candidate by integrating information.

The success of the particle filter relies mainly on three factors. First, in the particle prediction step, the Volterra kernels allow the particles to make reasonably accurate proposals based on the observed neural activities. This gives a good measure for evaluating the fitness of each particle. Second, in the resampling step, the weight of each particle embodies all the earlier observations, and because our particle filter keeps track of all proposals within the last 200 ms, earlier hypotheses can continue to be reevaluated and refined. Finally, in the propagation step, the particle filter utilizes prior knowledge about the manner in which the stimulus moves. This helps further in pruning down the hypothesis space.

Acknowledgments

This research is supported by NSF CAREER 9984706, NIH Vision Research core grant EY08098, and a NIH 2P41PR06009-11 for biomedical supercomputing. Thanks to Rick Romero, Yuguo Yu, and Anthony Brockwell for helpful discussion and advice.

# References

[1] A. E. Brockwell, A. L. Rojas, and R. E. Kass. Bayesian decoding of motor cortical signals by particle filtering. Submitted to J. Neurophysiology, 2003.

[2] E. Brown, L. Frank, D. Tang, M. Quirk, and M. Wilson. A statistical paradigm for neural spike train decoding applied to position prediction from ensemble firing patterns of rat hippocampal place cells. *J. Neuroscience*, 18(18):7411–7425, 1998.

[3] U.T. Eden, L.M. Frank, R. Barbieri, and E.N. Brown. Particle filtering algorithms for neural decoding and adaptive estimation of receptive field plasticity. In *Proc. Computational Neuroscience Meeting, CNS '02*, Santa Barbara, 2002.

[4] Y. Gao, M. J. Black, E. Bienenstock, S. Shoham, and J. P. Donoghue. *Probabilistic Inference of Hand Motion from Neural Activity in Motor Cortex*, pages 213–220. MIT Press, Cambridge, MA, 2002.

[5] A. P. Georgopoulos, A. B. Schwartz, and R. E. Kettner. Neuronal population coding of movement direction. *Science*, 243:234–236, 1989.

[6] F. Rieke, D. Warland, R. deRuytervanSteveninck, and W. Bialek. *Spikes: Exploring the Neural Code*. MIT Press, Cambridge, MA, 1997.

[7] R. Romero, Y. Yu, P Afhsar, and T. S. Lee. Adaptation of the temporal receptive fields of macaque v1 neurons. *Neurocomputing*, 52-54:135–140, 2002.

[8] G. Stanley, F. Li, and Y. Dan. Reconstruction of natural scenes from ensemble responses in the lateral geniculate nucleus. *J. Neuroscience*, 19(18):8036–8042, 1999.

[9] G. B. Stanley. Adaptive spatiotemporal receptive field estimation in the visual pathway. *Neural Computation*, 14:2925–2946, 2002.

[10] K. Zhang, I. Ginzburg, B.L. McNaughton, and T. J. Sejnowski. Interpreting neuronal population activity by reconstruction: Unified framework with application to hippocampal place cells. *J. Neurophysiology*, 79:1017–1044, 1998.
